# JANUS: Speech-to-Speech Translation Using Connectionist and Non-Connectionist Techniques

**Alex Waibel*** **Ajay N. Jain†**
**Arthur McNair** **Joe Tebelskis**
School of Computer Science
Carnegie Mellon University
Pittsburgh, PA 15213

**Louise Osterholtz**
Computational Linguistics Program
Carnegie Mellon University

**Hiroaki Saito**
Keio University
Tokyo, Japan

**Otto Schmidbauer**
Siemens Corporation
Munich, Germany

**Tilo Sloboda** **Monika Woszczyna**
University of Karlsruhe
Karlsruhe, Germany

## ABSTRACT

We present JANUS, a speech-to-speech translation system that utilizes diverse processing strategies, including connectionist learning, traditional AI knowledge representation approaches, dynamic programming, and stochastic techniques. JANUS translates continuously spoken English and German into German, English, and Japanese. JANUS currently achieves 87% translation fidelity from English speech and 97% from German speech. We present the JANUS system along with comparative evaluations of its interchangeable processing components, with special emphasis on the connectionist modules.

*Also with University of Karlsruhe, Karlsruhe, Germany.

†Now with Alliant Techsystems Research and Technology Center, Hopkins, Minnesota.

# 1  INTRODUCTION

In an age of increasing globalization of our economies and ever more efficient communication media, one important challenge is the need for effective ways of overcoming language barriers. Human translation efforts are generally expensive and slow, thus eliminating this possibility between individuals and around rapidly changing material (e.g. newscasts, newspapers). This need has recently lead to a resurgence of effort in machine translation—mostly of written language.

Much of human communication, however, is spoken, and the problem of spoken language translation must also be addressed. If successful, speech-to-text translation systems could lead to automatic subtitles in TV-broadcasts and cross-linguistic dictation. Speech-to-speech translation could be deployed as interpreting telephone service in restricted domains such as cross-linguistic hotel/conference reservations, catalog purchasing, travel planning, etc., and eventually in general domains, such as person-to-person telephone calls. Apart from telephone service, speech translation could facilitate multilingual negotiations and collaboration in face-to-face or video-conferencing settings.

With the potential applications so promising, what are the scientific challenges? Speech translation systems will need to address three distinct problems:

- *Speech Recognition and Understanding*: A naturally spoken utterance must be recognized and understood in the context of ongoing dialog.
- *Machine Translation*: A recognized message must be translated from one language into another (or several others).
- *Speech Synthesis*: A translated message must be synthesized in the target language.

Considerable challenges still face the development of each of the components, let alone the combination of the three. Among them only speech synthesis is mature enough for commercial systems to exist that can synthesize intelligible speech in several languages from text. But even here, to guarantee acceptance of the translation system, research is needed to improve naturalness and to allow for adaptation of the output speech (in the target language) to the voice characteristics of the input speaker. Speech recognition systems to date are generally limited in vocabulary size, and can only accept grammatically well-formed utterances. They require improvement to handle spontaneous unrestricted dialogs. Machine Translation systems require considerable development effort to work in a given language pair and domain reasonably well, and generally require syntactically well-formed input sentences. Improvements are needed to handle ill-formed sentences well and to allow for flexibility in the face of changes in domain and language pairs.

Beyond the challenges facing each system component, the combination of the three also introduces extra difficulties. Both the speech recognition and machine translation components, must deal with *spoken* language—ill-formed noisy input, both acoustically as well as syntactically. Therefore, the speech recognition component must be concerned less with *transcription* fidelity than *semantic* fidelity, while the MT-component must try to capture the meaning or intent of the input sentence without being guaranteed a syntactically legal sequence of words. In addition, non-symbolic prosodic information (intonation, rhythm, etc.) and dialog state must be taken into consideration to properly translate an input utterance. A closer cooperation between traditional signal processing and language level processing must be achieved.

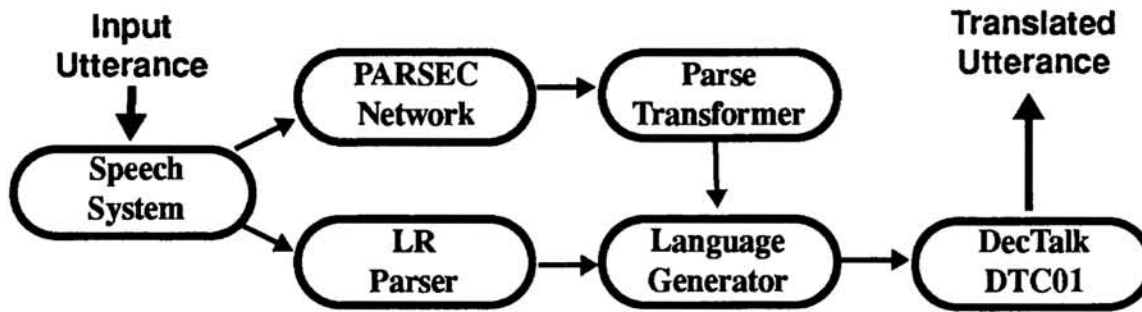

Figure 1: High-level JANUS architecture

JANUS is our first attempt at multilingual speech translation. It is the result of a collaborative effort between ATR Interpreting Telephony Research Laboratories, Carnegie Mellon University, Siemens Corporation, and the University of Karlsruhe. JANUS currently accepts continuously spoken sentences from a conference registration scenario, where a fictitious caller attempts to register to an international conference. The dialogs are read aloud from dialog scripts that make use of a vocabulary of approximately 400 words. Speaker-dependent and independent versions of the input recognition systems have been developed. JANUS currently accepts continuously spoken English and German input and produces spoken German, English, and Japanese output as a result.

While JANUS has some of the limitations mentioned above, it is the first tri-lingual continuous large vocabulary speech translation system to-date. It is a vehicle toward overcoming some of the limitations described. A particular focus is the trainability of system components, so that flexible, adaptive, and robust systems may result. JANUS is a hybrid system that uses a blend of computational strategies: connectionist, statistical and knowledge based techniques. This paper will describe each of JANUS's processing components separately and particularly highlight the relative contributions of connectionist techniques within this ensemble. Figure 1 shows a high-level diagram of JANUS's components.

## 2  SPEECH RECOGNITION

Two alternative speech recognition systems are currently used in JANUS: Linked Predictive Neural Networks (LPNNs) and Learned Vector Quantization networks (LVQ) (Tebelskis *et al.* 1991; Schmidbauer and Tebelskis 1992). They are both connectionist, continuous-speech recognition systems, and both have vocabularies of approximately 400 English and 400 German words. Each use statistical bigram or word-pair grammars derived from the conference registration database. The systems are based on canonical phoneme models (states) that can be logically concatenated in any order to create models for different words. The need for training data with labeled phonemes can be reduced by first bootstrapping the networks on a small amount of speech with forced *phoneme* boundaries, then training on the whole database using only forced *word* boundaries.

In the LPNN system, each phoneme model is implemented by a predictive neural network. Each network is trained to accurately predict the next frame of speech within segments of speech corresponding to its phoneme model. Continuous scores (prediction errors) are accumulated for various word candidates. The LPNN module produces either a single

hypothesized sentence or the first N best hypotheses using a modified dynamic-programming beam-search algorithm (Steinbiss 1989). The LPNN system has speaker-dependent word accuracy rates of 93% with first-best recognition, and sentence accuracy of 69%.

LVQ is a vector clustering technique based on neural networks. We have used LVQ to automatically cluster speech frames into a set of acoustic features; these features are fed into a set of output units that compute the emission probability for HMM states. This technique gives speaker-dependent word accuracy rates of 98%, 86%, and 82% for English conference registration tasks of perplexity 7, 61, and 111, respectively. The sentence recognition rate at perplexity 7 is 80%.

We are also evaluating other approaches to speech recognition, such as the Multi-State TDNN for continuous-speech (Haffner, Franzini, and Waibel 1991) and a neural-network based word spotting system that may be useful for modeling spontaneous speech effects (Zeppenfield and Waibel 1992). The recognitions systems' text output serves as input to the alternative parsing modules of JANUS.

# 3  LANGUAGE UNDERSTANDING AND TRANSLATION

## 3.1 LANGUAGE ANALYSIS

The translation module of JANUS is based on the Universal Parser Architecture (UPA) developed at Carnegie Mellon (Tomita and Carbonell 1987; Tomita and Nyberg 1988). It is designed for efficient multi-lingual translation. Text in a source language is parsed into a language independent frame-based *interlingual* representation. From the interlingua, text can be generated in different languages.

The system requires hand-written parsing and generation grammars for each language to be processed. The parsing grammars are based on a Lexical Functional Grammar formalism, and are implemented using Tomita's Generalized LR parsing Algorithm (Tomita 1991). The generation grammars are compiled into LISP functions. Both parsing and generation with UPA approach real-time. Figure 2 shows an example of the input, interlingual representation, and the output of the JANUS system

## 3.2 PARSEC: CONNECTIONIST PARSING

JANUS can use a connectionist parser in place of the LR parser to process the output of the speech system. PARSEC is a structured connectionist parsing architecture that is geared toward the problems found in spoken language (for details, see Jain 1992 (in this volume) and Jain's PhD thesis, in preparation). PARSEC networks exhibit three strengths:

- They automatically learn to parse, and generalize well compared to hand-coded grammars.
- They tolerate several types of noise without any explicit noise modeling.
- They can learn to use multi-modal input such as pitch in conjunction with syntax and semantics.

The PARSEC network architecture relies on a variation of supervised back-propagation learning. The architecture differs from some other connectionist approaches in that it is highly structured, both at the macroscopic level of modules, and at the microscopic level of connections.

## Input

Hello is this the office for the conference.

## Interlingual Representation

```
((CFNAME *is-this-phone)
 (MOOD *interrogative)
 (OBJECT ((NUMBER sg) (DET the)
          (CFNAME *conf-office)))
 (SADJUNCT1 ((CFNAME *hello))))
```

## Output

Japanese: MOSHI MOSHI KAIGI JIMUKYOKU DESUKA
German: HALLO IST DIES DAS KONFERENZBUERO

Figure 2: Example of input, interlingua, and output of JANUS

### 3.2.1 Learning and Generalization

Through exposure to example output parses, PARSEC networks learn parsing behavior. Trained networks generalize well compared to hand-written grammars. In direct tests of coverage for the conference registration domain, PARSEC achieved 67% correct parsing of novel sentences, whereas hand-written grammars achieved just 5%, 25%, and 38% correct. Two of the grammars were written as part of a contest with a large cash prize for best coverage.

The process of training PARSEC networks is highly automated, and is made possible through the use of constructive learning coupled with a robust control procedure that dynamically adjusts learning parameters during training. Novice users of the PARSEC system were able to train networks for parsing a German-language version of the conference registration task and a novel English air-travel reservation task.

### 3.2.2 Noise Tolerance

We have compared PARSEC's performance on noisy input with that of hand-written grammars. On synthetic ungrammatical conference registration sentences, PARSEC produced acceptable interpretations 66% of the time, with the three hand-coded grammars mentioned above performing at 2%, 38%, and 34%, respectively. We have also evaluated PARSEC in the context of noisy speech recognition in JANUS, and this is discussed later.

### 3.2.3 Multi-Modal Input

A somewhat elusive goal of spoken language processing has been to utilize information from the speech signal beyond just word sequences in higher-level processing. It is well known that humans use such information extensively in conversation. Consider the utterances "Okay." and "Okay?" Although semantically distinct, they cannot be distinguished based on word sequence, but pitch contours contain the necessary information (Figure 3).

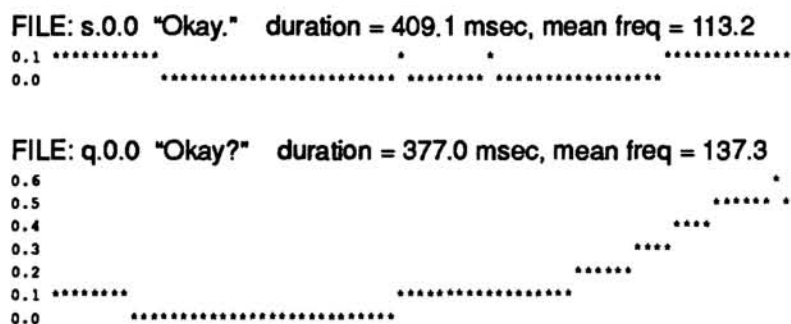

Figure 3: Smoothed pitch contours.

In a grammar-based system, it is difficult to incorporate real-valued vector input in a useful way. In a PARSEC network, the vector is just another set of input units. A module of a PARSEC network was augmented to contain an additional set of units that contained pitch information. The pitch contours were smoothed output from the OGI Neural Network Pitch Tracker (Barnard *et al.* 1991).

Within the JANUS system, the augmented PARSEC network brings new functionality. Intonation affects translation in JANUS when using the augmented PARSEC network. The sentence, "This is the conference office." is translated to "Kaigi jimukyoku desu." "This is the conference office?" is translated to "Kaigi jimukyoku desuka?" This required no changes in the other modules of the JANUS system. It also should be possible to use other types of information from the speech signal to aid in robust parsing (e.g. energy patterns to disambiguate clausal structure).

## 4  SPEECH SYNTHESIS

To generate intelligible speech in the respective target languages, we have predominantly used commercial devices. Most notably, DEC-talk has provided unrestricted English text-to-speech synthesis. DEC-talk has also been used for Japanese and German synthesis. The internal English text-to-phoneme conversion rules and tables of DEC-talk were bypassed by external German and Japanese text-to-phoneme look-up tables that convert the German/Japanese target sentences into phonemic strings for DEC-talk synthesis. The resulting synthesis is limited to the task vocabulary, but the external tables result in intelligible German and Japanese speech—albeit with a pronounced American accent.

To allow for greater flexibility in vocabulary and more language specific synthesis, several alternate devices are currently being integrated. For Japanese, in particular, two high quality speech synthesizers developed separately by NEC and ATR will be used to provide more satisfactory results. In JANUS, no attempt has so far been made to adapt the output speech to the input speaker's voice characteristics. However, this has recently been demonstrated by work with code book mapping (Abe, Shikano, and Kuwabara 1990) and connectionist mapping techniques (Huang, Lee, and Waibel 1991).

# 5  IMPLEMENTATION ISSUES AND PERFORMANCE

## 5.1  Parallel Hardware

Neural network forward passes for the speech recognizer were programmed on two general purpose parallel machines, a MasPar computer at the University of Karlsruhe, Germany and an Intel iWarp at Carnegie Mellon. The MasPar is a parallel SIMD machine with 4096 processing elements. The iWarp is a MIMD machine, and a 16MHz, 64 cell experimental version was used for testing.

The use of parallel hardware and algorithms has significantly decreased JANUS's processing time. Compared to forward pass calculations performed by a DecStation 5000, the iWarp is 9 times faster (41.4 million connections per second). The MasPar does the forward pass calculations for a two second utterance in less than 500 milliseconds. Both the iWarp and MasPar are scalable. Efforts are underway to implement other parts of JANUS on parallel hardware with the goal of near real-time performance.

## 5.2  Performance

Currently, English JANUS using the LR parsing module (JANUS-LR) performs at 87% correct translation using the LPNN speech system with the N-best sentence hypotheses. German JANUS performs at 97% correct translation (on a subset of the conference registration database) using German versions of the LPNN system and LR parsing grammar.

English JANUS using PARSEC (JANUS-NN) does not perform as well as the LR parser version in N-best mode, with 80% correct translation. PARSEC is not able to select from a list of ranked candidate utterance hypotheses as robustly as is the LR parser using a very tight grammar. However, the grammar used for this comparison only achieves 5% coverage of novel test sentences, compared with PARSEC's 67%. This vast difference in coverage explains some of the N-best performance difference.

In First-best mode, however, JANUS-NN does *better* than JANUS-LR (77% versus 70%). The PARSEC network is able to produce acceptable parses for a number of noisy speech recognition hypotheses, but JANUS-LR tends to reject those hypotheses as unparsable. PARSEC's flexibility, which hurt its N-best performance, enhances its F-best performance. No performance evaluations were carried out using German PARSEC in German JANUS.

# 6  CONCLUSION

In this paper we have described JANUS, a multi-lingual speech-to-speech translation system. JANUS uses a mixture of connectionist, statistical and rule based strategies to achieve this goal. Connectionist models have contributed in providing high performance recognition and parsing performance as well as greater robustness in the light of task variations and syntactically ill-formed sentences. Connectionist models also provide a mechanism for merging traditionally distinct symbolic (syntax) and signal-level (intonation) information gracefully and achieve successful disambiguation between grammatical statements whose mood can be affected by intonation. Finally, connectionist sentence analysis appears to offer high flexibility as the relevant modules can be retrained automatically for new tasks, domains and even languages without laborious recoding. We plan to continue exploring different mixtures of computing paradigms to achieve higher performance.

## Acknowledgements

The authors gratefully acknowledge the support of ATR Interpreting Telephony Laboratories, Siemens Corporation, NEC Corporation, and the National Science Foundation.

## References

Abe, M., K. Shikano, H. Kuwabara. 1990. Cross Language Voice Conversion. In *IEEE Proceedings of the International Conference on Acoustics, Speech, and Signal Processing*.

Barnard, E., R. A. Cole, M. P. Vea, F. A. Alleva. 1991. Pitch Detection with a Neural-Net Classifier. *IEEE Transactions on Signal Processing* 39(2): 298–307.

Haffner, P., M. Franzini, and A. Waibel. 1991. Integrating time alignment and neural networks for high performance speech recognition. In *IEEE Proceedings of the International Conference on Acoustics, Speech, and Signal Processing*.

Huang, X. D., K. F. Lee, A. Waibel. 1991. In *Proceedings of the IEEE-SP Workshop on Neural Networks for Signal Processing*.

Jain, A. N. 1992. Generalization performance in PARSEC—A structured connectionist learning architecture. In *Advances in Neural Information Processing Systems 4*, ed. J. E. Moody, S. J. Hanson, and R. P. Lippmann. San Mateo, CA: Morgan Kaufmann Publishers.

Jain, A. N. In preparation. PARSEC: A Connectionist Learning Architecture for Parsing Spoken Language. PhD Thesis, School of Computer Science, Carnegie Mellon University.

Schmidbauer, O. and J. Tebelskis. 1992. An LVQ based reference model for speaker-adaptive speech recognition. In *IEEE Proceedings of the International Conference on Acoustics, Speech, and Signal Processing*.

Steinbiss, V. 1989. Sentence-hypothesis generation in a continuous-speech recognition system. In *Proceedings of the 1989 European Conference on Speech Communication and Technology*, Vol. 2, 51–54.

Tebelskis, J., A. Waibel, B. Petek, O. Schmidbauer. 1991. Continuous speech recognition by Linked Predictive Neural Networks. In *Advances in Neural Information Processing System 3*, ed. R. Lippmann, J. Moody, and D. Touretzky. San Mateo, CA: Morgan Kaufmann Publishers.

Tomita, M. (ed.). 1991. *Generalized LR Parsing*. Norwell, MA: Kluwer Academic Publishers.

Tomita, M. and J. G. Carbonell. 1987. *The Universal Parser Architecture for Knowledge-Based Machine Translation*. Technical Report CMU-CMT-87-01, Center for Machine Translation, Carnegie Mellon University.

Tomita, M. and E. Nyberg. 1988. *Generation Kit and Transformation Kit*. Technical Report CMU-CMT-88-MEMO, Center for Machine Translation, Carnegie Mellon University.

Zeppenfield, T. and A. Waibel. 1992. A hybrid neural network, dynamic programming word spotter. In *IEEE Proceedings of the International Conference on Acoustics, Speech, and Signal Processing*.